# Sequential Bayesian Kernel Regression

**Jaco Vermaak, Simon J. Godsill, Arnaud Doucet**
Cambridge University Engineering Department
Cambridge, CB2 1PZ, U.K.
{jv211, sjg, ad2}@eng.cam.ac.uk

## Abstract

We propose a method for sequential Bayesian kernel regression. As is the case for the popular Relevance Vector Machine (RVM) [10, 11], the method automatically identifies the number and locations of the kernels. Our algorithm overcomes some of the computational difficulties related to batch methods for kernel regression. It is non-iterative, and requires only a single pass over the data. It is thus applicable to truly sequential data sets and batch data sets alike. The algorithm is based on a generalisation of Importance Sampling, which allows the design of intuitively simple and efficient proposal distributions for the model parameters. Comparative results on two standard data sets show our algorithm to compare favourably with existing batch estimation strategies.

## 1 Introduction

Bayesian kernel methods, including the popular Relevance Vector Machine (RVM) [10, 11], have proved to be effective tools for regression and classification. For the RVM the sparsity constraints are elegantly formulated within a Bayesian framework, and the result of the estimation is a mixture of kernel functions that rely on only a small fraction of the data points. In this sense it bears resemblance to the popular Support Vector Machine (SVM) [13]. Contrary to the SVM, where the support vectors lie on the decision boundaries, the relevance vectors are prototypical of the data. Furthermore, the RVM does not require any constraints on the types of kernel functions, and provides a probabilistic output, rather than a hard decision.

Standard batch methods for kernel regression suffer from a computational drawback in that they are iterative in nature, with a computational complexity that is normally cubic in the number of data points at each iteration. A large proportion of the research effort in this area is devoted to the development of estimation algorithms with reduced computational complexity. For the RVM, for example, a strategy is proposed in [12] that exploits the structure of the marginal likelihood function to significantly reduce the number of computations.

In this paper we propose a full Bayesian formulation for kernel regression on sequential data. Our algorithm is non-iterative, and requires only a single pass over the data. It is equally applicable to batch data sets by presenting the data points one at a time, with the order of presentation being unimportant. The algorithm is especially effective for large data sets. As opposed to batch strategies that attempt to find the optimal solution conditional on all the data, the sequential strategy includes the data one at a time, so that the poste-

rior exhibits a tempering effect as the amount of data increases. Thus, the difficult global estimation problem is effectively decomposed into a series of easier estimation problems.

The algorithm itself is based on a generalisation of Importance Sampling, and recursively updates a sample based approximation of the posterior distribution as more data points become available. The proposal distribution is defined on an augmented parameter space, and is formulated in terms of model moves, reminiscent of the Reversible Jump Markov Chain Monte Carlo (RJ-MCMC) algorithm [5]. For kernel regression these moves may include update moves to refine the kernel locations, birth moves to add new kernels to better explain the increasing data, and death moves to eliminate erroneous or redundant kernels.

The remainder of the paper is organised as follows. In Section 2 we outline the details of the model for sequential Bayesian kernel regression. In Section 3 we present the sequential estimation algorithm. Although we focus on regression, the method extends straightforwardly to classification. It can, in fact, be applied to any model for which the posterior can be evaluated up to a normalising constant. We illustrate the performance of the algorithm on two standard regression data sets in Section 4, before concluding with some remarks in Section 5.

## 2 Model Description

The data is assumed to arrive sequentially as input-output pairs $(\mathbf{x}_t, y_t)$, $t = 1, 2, \cdots$, $\mathbf{x}_t \in \mathbb{R}^d$, $y_t \in \mathbb{R}$. For kernel regression the output is assumed to follow the model

$$y_t = \beta_0 + \sum_{i=1}^{k} \beta_i K(\mathbf{x}_t, \boldsymbol{\mu}_i) + v_t, \quad v_t \sim \mathcal{N}(0, \sigma_y^2),$$

where $k$ is the number of kernel functions, which we will consider to be unknown, $\boldsymbol{\beta}_k = (\beta_0 \cdots \beta_k)$ are the regression coefficients, $\mathbf{U}_k = (\boldsymbol{\mu}_1 \cdots \boldsymbol{\mu}_k)$ are the kernel centres, and $\sigma_y^2$ is the variance of the Gaussian observation noise. Assuming independence the likelihood for all the data points observed up to time $t$, denoted by $\mathbf{Y}_t = (y_1 \cdots y_t)$, can be written as

$$p(\mathbf{Y}_t | k, \boldsymbol{\beta}_k, \mathbf{U}_k, \sigma_y^2) = \mathcal{N}(\mathbf{Y}_t | \mathbf{K}_k \boldsymbol{\beta}_k, \sigma_y^2 \mathbf{I}_t), \tag{1}$$

where $\mathbf{K}_k$ denotes the $t \times (k + 1)$ kernel matrix with $[\mathbf{K}_k]_{s,1} = 1$ and $[\mathbf{K}_k]_{s,l} = K(\mathbf{x}_s, \boldsymbol{\mu}_{l-1})$ for $l > 1$, and $\mathbf{I}_n$ denotes the $n$-dimensional identity matrix. For the unknown model parameters $\boldsymbol{\theta}_k = (\boldsymbol{\beta}_k, \mathbf{U}_k, \sigma_y^2, \sigma_\beta^2)$ we assume a hierarchical prior that takes the form

$$p(k, \boldsymbol{\theta}_k) = p(k)p(\boldsymbol{\beta}_k, \sigma_\beta^2)p(\mathbf{U}_k)p(\sigma_y^2), \tag{2}$$

with

$$p(k) \propto \lambda^k \exp(-\lambda)/k!, \quad k \in \{1 \cdots k_{\max}\}$$

$$p(\boldsymbol{\beta}_k, \sigma_\beta^2) = \mathcal{N}(\boldsymbol{\beta}_k | \mathbf{0}, \sigma_\beta^2 \mathbf{I}_{k+1}) \mathcal{IG}(\sigma_\beta^2 | a_\beta, b_\beta)$$

$$p(\mathbf{U}_k) = \prod_{l=1}^{k} \sum_{s=1}^{t} \delta_{\mathbf{x}_s}(\boldsymbol{\mu}_l)/t$$

$$p(\sigma_y^2) = \mathcal{IG}(\sigma_y^2 | a_y, b_y),$$

where $\delta_x(\cdot)$ denotes the Dirac delta function with mass at $x$, and $\mathcal{IG}(\cdot | a, b)$ denotes the Inverted Gamma distribution with parameters $a$ and $b$. The prior on the number of kernels is set to be a truncated Poisson distribution, with the mean $\lambda$ and the maximum number of kernels $k_{\max}$ assumed to be fixed and known. The regression coefficients are drawn from an isotropic Gaussian prior with variance $\sigma_\beta^2$ in each direction. This variance is, in turn, drawn from an Inverted Gamma prior. This is in contrast with the Automatic Relevance Determination (ARD) prior [8], where each coefficient has its own associated variance. The prior for the kernel centres is assumed to be uniform over the grid formed by the input

data points available at the current time step. Note that the support for this prior increases with time. Finally, the noise variance is assumed to follow an Inverted Gamma prior. The parameters of the Inverted Gamma priors are assumed to be fixed and known.

Given the likelihood and prior in (1) and (2), respectively, it is straightforward to obtain an expression for the full posterior distribution $p(k, \boldsymbol{\theta}_k | \mathbf{Y}_t)$. Due to conjugacy this expression can be marginalised over the regression coefficients, so that the marginal posterior for the kernel centres can be written as

$$p(k, \mathbf{U}_k | \sigma_y^2, \sigma_\beta^2, \mathbf{Y}_t) \propto \frac{|\mathbf{B}_k|^{1/2} \exp(-\mathbf{Y}_t^\mathsf{T} \mathbf{P}_k \mathbf{Y}_t / 2\sigma_y^2) p(k) p(\mathbf{U}_k)}{(2\pi\sigma_y^2)^{t/2} (\sigma_\beta^2)^{k+1/2}}, \qquad (3)$$

with $\mathbf{B}_k = (\mathbf{K}_k^\mathsf{T} \mathbf{K}_k / \sigma_y^2 + \mathbf{I}_{k+1} / \sigma_\beta^2)^{-1}$ and $\mathbf{P}_k = \mathbf{I}_t - \mathbf{K}_k \mathbf{B}_k \mathbf{K}_k^\mathsf{T} / \sigma_y^2$. It will be our objective to approximate this distribution recursively in time as more data becomes available, using Monte Carlo techniques. Once we have samples for the kernel centres, we will require new samples for the unknown parameters $(\sigma_y^2, \sigma_\beta^2)$ at the next time step. We can obtain these by first sampling for the regression coefficients from the posterior

$$p(\boldsymbol{\beta}_k | k, \mathbf{U}_k, \sigma_y^2, \sigma_\beta^2, \mathbf{Y}_t) = \mathcal{N}(\boldsymbol{\beta}_k | \widehat{\boldsymbol{\beta}}_k, \mathbf{B}_k), \qquad (4)$$

with $\widehat{\boldsymbol{\beta}}_k = \mathbf{B}_k \mathbf{K}_k^\mathsf{T} \mathbf{Y}_t$, and conditional on these values, sampling for the unknown parameters from the posteriors

$$\begin{aligned} p(\sigma_y^2 | k, \boldsymbol{\beta}_k, \mathbf{U}_k, \mathbf{Y}_t) &= \mathcal{IG}(\sigma_y^2 | a_y + t/2, b_y + \mathbf{e}_t^\mathsf{T} \mathbf{e}_t / 2) \\ p(\sigma_\beta^2 | k, \boldsymbol{\beta}_k) &= \mathcal{IG}(\sigma_\beta^2 | a_\beta + (k+1)/2, b_\beta + \boldsymbol{\beta}_k^\mathsf{T} \boldsymbol{\beta}_k / 2), \end{aligned} \qquad (5)$$

with $\mathbf{e}_t = \mathbf{Y}_t - \mathbf{K}_k \boldsymbol{\beta}_k$ the model approximation error.

Since the number of kernel functions to use is unknown the marginal posterior in (3) is defined over a discrete space of variable dimension. In the next section we will present a generalised importance sampling strategy to obtain Monte Carlo approximations for distributions of this nature recursively as more data becomes available.

## 3   Sequential Estimation

Recall that it is our objective to recursively update a Monte Carlo representation of the posterior distribution for the kernel regression parameters as more data becomes available. The method we propose here is based on a generalisation of the popular importance sampling technique. Its application extends to any model for which the posterior can be evaluated up to a normalising constant. We will thus first present the general strategy, before outlining the details for sequential kernel regression.

### 3.1   Generalised Importance Sampling

Our aim is to recursively update a sample based approximation of the posterior $p(k, \boldsymbol{\theta}_k | \mathbf{Y}_t)$ of a model parameterised by $\boldsymbol{\theta}_k$ as more data becomes available. The efficiency of importance sampling hinges on the ability to design a good proposal distribution, *i.e.* one that approximates the target distribution sufficiently well. Designing an efficient proposal distribution to generate samples directly in the target parameter space is difficult. This is mostly due to the fact that the dimension of the parameter space is generally high and variable. To circumvent these problems we augment the target parameter space with an auxiliary parameter space, which we will associate with the parameters at the previous time step. We now define the target distribution over the resulting joint space as

$$\pi_t(k, \boldsymbol{\theta}_k; k', \boldsymbol{\theta}_{k'}') = p(k, \boldsymbol{\theta}_k | \mathbf{Y}_t) q_t'(k', \boldsymbol{\theta}_{k'}' | k, \boldsymbol{\theta}_k). \qquad (6)$$

This joint clearly admits the desired target distribution as a marginal. Apart from some weak assumptions, which we will discuss shortly, the distribution $q_t'$ is entirely arbitrary, and may depend on the data and the time step. In fact, in the application to the RVM we consider here we will set it to $q_t'(k', \boldsymbol{\theta}_{k'}'|k, \boldsymbol{\theta}_k) = \delta_{(k, \boldsymbol{\theta}_k)}(k', \boldsymbol{\theta}_{k'}')$, so that it effectively disappears from the expression above. A similar strategy of augmenting the space to simplify the importance sampling procedure has been exploited before in [7] to develop efficient Sequential Monte Carlo (SMC) samplers for a wide range of models. To generate samples in this joint space we define the proposal for importance sampling to be of the form

$$Q_t(k, \boldsymbol{\theta}_k; k', \boldsymbol{\theta}_{k'}') = p(k', \boldsymbol{\theta}_{k'}'|\mathbf{Y}_{t-1})q_t(k, \boldsymbol{\theta}_k|k', \boldsymbol{\theta}_{k'}'), \qquad (7)$$

where $q_t$ may again depend on the data and the time step. This proposal embodies the sequential character of our algorithm. Similar to SMC methods [3] it generates samples for the parameters at the current time step by incrementally refining the posterior at the previous time step through the distribution $q_t$. Designing efficient incremental proposals is much easier than constructing proposals that generate samples directly in the target parameter space, since the posterior is unlikely to undergo dramatic changes over consecutive time steps. To compensate for the discrepancy between the proposal in (7) and the joint posterior in (6) the importance weight takes the form

$$W_t(k, \boldsymbol{\theta}_k; k', \boldsymbol{\theta}_{k'}') = \frac{p(k, \boldsymbol{\theta}_k|\mathbf{Y}_t)q_t'(k', \boldsymbol{\theta}_{k'}'|k, \boldsymbol{\theta}_k)}{p(k', \boldsymbol{\theta}_{k'}'|\mathbf{Y}_{t-1})q_t(k, \boldsymbol{\theta}_k|k', \boldsymbol{\theta}_{k'}')}. \qquad (8)$$

Due to the construction of the joint in (6), marginal samples in the target parameter space associated with this weighting will indeed be distributed according to the target posterior $p(k, \boldsymbol{\theta}_k|\mathbf{Y}_t)$. As might be expected the importance weight in (8) is similar in form to the acceptance ratio for the RJ-MCMC algorithm [5]. One notable difference is that the reversibility condition is not required, so that for a given $q_t$, $q_t'$ may be arbitrary, as long as the ratio in (8) is well-defined.

In practice it is often necessary to design a number of candidate moves to obtain an efficient algorithm. Examples include update moves to refine the model parameters in the light of the new data, birth moves to add new parameters to better explain the new data, death moves to remove redundant or erroneous parameters, and many more. We will denote the set of candidate moves at time $t$ by $\{\alpha_{t,i}, q_{t,i}, q_{t,i}'\}_{i=1}^M$, where $\alpha_{t,i}$ is the probability of choosing move $i$, with $\sum_{i=1}^M \alpha_{t,i} = 1$. For each move $i$ the importance weight is computed by substituting the corresponding $q_{t,i}$ and $q_{t,i}'$ into (8). Note that the probability of choosing a particular move may depend on the old state and the time step, so that moves may be included or excluded as is appropriate.

## 3.2 Sequential Kernel Regression

We will now present the details for sequential kernel regression. Our main concern will be the recursive estimation of the marginal posterior for the kernel centres in (3). This distribution is conditional on the parameters $(\sigma_y^2, \sigma_\beta^2)$, for which samples can be obtained at each time step from the corresponding posteriors in (4) and (5).

To sample for the new kernel centres we will consider three kinds of moves: a zero move $q_{t,1}$, a birth move $q_{t,2}$, and a death move $q_{t,3}$. The zero move leaves the kernel centres unchanged. The birth move adds a new kernel at a uniformly randomly chosen location over the grid of unoccupied input data points. The death move removes a uniformly randomly chosen kernel. For $k = 0$ only the birth move is possible, whereas the birth move is impossible for $k = k_{\max}$ or $k = t$. Similar to [5] we set the move probabilities to

$$\alpha_{t,2} = c \min\{1, p(k+1)/p(k)\}$$
$$\alpha_{t,3} = c \min\{1, p(k-1)/p(k)\}$$
$$\alpha_{t,1} = 1 - \alpha_{t,2} - \alpha_{t,3}$$

in all other cases. In the above $c \in (0,1)$ is a parameter that tunes the relative frequency of the dimension changing moves to the zero move. For these choices the importance weight in (8) becomes

$$W_{t,i}(k, \mathbf{U}_k; k', \mathbf{U}'_{k'}) \propto \frac{|\mathbf{B}_k|^{1/2} \exp(-(\mathbf{Y}_t^\mathsf{T} \mathbf{P}_k \mathbf{Y}_t - \mathbf{Y}_{t-1}^\mathsf{T} \mathbf{P}'_{k'} \mathbf{Y}_{t-1})/2\sigma_y^2)}{|\mathbf{B}'_{k'}|^{1/2}(2\pi\sigma_y^2)^{1/2}(\sigma_\beta^2)^{k-k'/2}}$$
$$\times \frac{\lambda^{k-k'}(t-1)(k'-1)!}{t(k-1)! q_{t,i}(k, \mathbf{U}_k | k', \mathbf{U}'_{k'})},$$

where the primed variables are those corresponding to the posterior at time $t-1$. For the zero move the parameters are left unchanged, so that the expression for $q_{t,1}$ in the importance weight becomes unity. This is often a good move to choose, and captures the notion that the posterior rarely changes dramatically over consecutive time steps. For the birth move one new kernel is added, so that $k = k' + 1$. The centre for this kernel is uniformly randomly chosen from the grid of unoccupied input data points. This means that the expression for $q_{t,2}$ in the importance weight reduces to $1/(t - k')$, since there are $t - k'$ such data points. Similarly, the death move removes a uniformly randomly chosen kernel, so that $k = k' - 1$. In this case the expression for $q_{t,3}$ in the importance weight reduces to $1/k'$. It is straightforward to design numerous other moves, *e.g.* an update move that perturbs existing kernel centres. However, we found that the simple moves presented yield satisfactory results while keeping the computational complexity acceptable.

We conclude this section with a summary of the algorithm.

---

**Algorithm 1: Sequential Kernel Regression**

*Inputs*:

- Kernel function $K(\cdot, \cdot)$, model parameters $(\lambda, k_{\max}, a_y, b_y, a_\beta, b_\beta)$, fraction of dimension change moves $c$, number of samples to approximate the posterior $N$.

*Initialisation*: $t = 0$

- For $i = 1 \cdots N$, set $k^{(i)} = 0$, $\beta_k^{(i)} = \emptyset$, $\mathbf{U}_k^{(i)} = \emptyset$, and sample $\sigma_y^{2(i)} \sim p(\sigma_y^2)$, $\sigma_\beta^{2(i)} \sim p(\sigma_\beta^2)$.

*Generalised Importance Sampling Step*: $t > 0$

- For $i = 1 \cdots N$,

    - Sample a move $j(i)$ so that $P(j(i) = l) = \alpha_{t,l}$.
    - If $j(i) = 1$ (zero move), set $\widetilde{\mathbf{U}}_k^{(i)} = \mathbf{U}_k^{(i)}$ and $\widetilde{k}^{(i)} = k^{(i)}$.
      Else if $j(i) = 2$ (birth move), form $\widetilde{\mathbf{U}}_k^{(i)}$ by uniformly randomly adding a kernel at one of the unoccupied data points, and set $\widetilde{k}^{(i)} = k^{(i)} + 1$.
      Else if $j(i) = 3$ (death move), form $\widetilde{\mathbf{U}}_k^{(i)}$ by uniformly randomly deleting one of the existing kernels, and set $\widetilde{k}^{(i)} = k^{(i)} - 1$.

- For $i = 1 \cdots N$, compute the importance weights $W_t^{(i)} \propto W_t(\widetilde{k}^{(i)}, \widetilde{\mathbf{U}}_k^{(i)}; k^{(i)}, \mathbf{U}_k^{(i)})$, and normalise.

- For $i = 1 \cdots N$, sample the nuisance parameters $\widetilde{\beta}_k^{(i)} \sim p(\beta_k | \widetilde{k}^{(i)}, \widetilde{\mathbf{U}}_k^{(i)}, \sigma_y^{2(i)}, \sigma_\beta^{2(i)}, \mathbf{Y}_t)$, $\widetilde{\sigma}_\beta^{2(i)} \sim p(\sigma_\beta^2 | \widetilde{k}^{(i)}, \widetilde{\beta}_k^{(i)})$, $\widetilde{\sigma}_y^{2(i)} \sim p(\sigma_y^2 | \widetilde{k}^{(i)}, \widetilde{\beta}_k^{(i)}, \widetilde{\mathbf{U}}_k^{(i)}, \mathbf{Y}_t)$.

*Resampling Step*: $t > 0$

- Multiply / discard samples $\{\widetilde{k}^{(i)}, \widetilde{\theta}_k^{(i)}\}$ with respect to high / low importance weights $\{W_t^{(i)}\}$ to obtain $N$ samples $\{k^{(i)}, \theta_k^{(i)}\}$.

■

---

Each of the samples is initialised to be empty, *i.e.* no kernels are included. Initial values for the variance parameters are sampled from their corresponding prior distributions. Using the samples before resampling, a Minimum Mean Square Error (MMSE) estimate of the clean data can be obtained as

$$\widehat{\mathbf{Z}}_t = \sum_{i=1}^{N} W_t^{(i)} \widetilde{\mathbf{K}}_k^{(i)} \widetilde{\boldsymbol{\beta}}_k^{(i)}.$$

The resampling step is required to avoid degeneracy of the sample based representation. It can be performed by standard procedures such as multinomial resampling [4], stratified resampling [6], or minimum entropy resampling [2]. All these schemes are unbiased, so that the number of times $N_i$ the sample $(\widetilde{k}^{(i)}, \widetilde{\boldsymbol{\theta}}_k^{(i)})$ appears after resampling satisfies $\mathbb{E}(N_i) = NW_t^{(i)}$. Thus, resampling essentially multiplies samples with high importance weights, and discards those with low importance weights.

The algorithm requires only a single pass through the data. The computational complexity at each time step is $O(N)$. For each sample the computations are dominated by the computation of the matrix $\mathbf{B}_k$, which requires a $(k+1)$-dimensional matrix inverse. However, this inverse can be incrementally updated from the inverse at the previous time step using the techniques described in [12], leading to substantial computational savings.

## 4 Experiments and Results

In this section we illustrate the performance of the proposed sequential estimation algorithm on two standard regression data sets.

### 4.1 Sinc Data

This experiment is described in [1]. The training data is taken to be the sinc function, *i.e.* $\text{sinc}(x) = \sin(x)/x$, corrupted by additive Gaussian noise of standard deviation $\sigma_y = 0.1$, for 50 evenly spaced points in the interval $x \in [-10, 10]$. In all the runs we presented these points to the sequential estimation algorithm in random order. For the test data we used 1000 points over the same interval. We used a Gaussian kernel of width 1.6, and set the fixed parameters of the model to $(\lambda, k_{\max}, a_y, b_y, a_\beta, b_\beta) = (1, 50, 0, 0, 0, 0)$. For these settings the prior on the variances reduces to the uninformative Jeffreys' prior. The fraction of dimension change moves was set to $c = 0.25$. It should be noted that the algorithm proved to be relatively insensitive to reasonable variations in the values of the fixed parameters.

The left side of Figure 1 shows the test error as a function of the number of samples $N$. These results were obtained by averaging over 25 random generations of the training data for each value of $N$. As expected, the error decreases with an increase in the number of samples. No significant decrease is obtained beyond $N = 250$, and we adopt this value for subsequent comparisons. A typical MMSE estimate of the clean data is shown on the right side of Figure 1.

In Table 1 we compare the results of the proposed sequential estimation algorithm with a number of batch strategies for the SVM and RVM. The results for the batch algorithms are duplicated from [1, 9]. The error for the sequential algorithm is slightly higher. This is due to the stochastic nature of the algorithm, and the fact that it uses only very simple moves that take no account of the characteristics of the data during the move proposition. This increase should be offset against the algorithm simplicity and efficiency. The error could be further decreased by designing more complex moves.

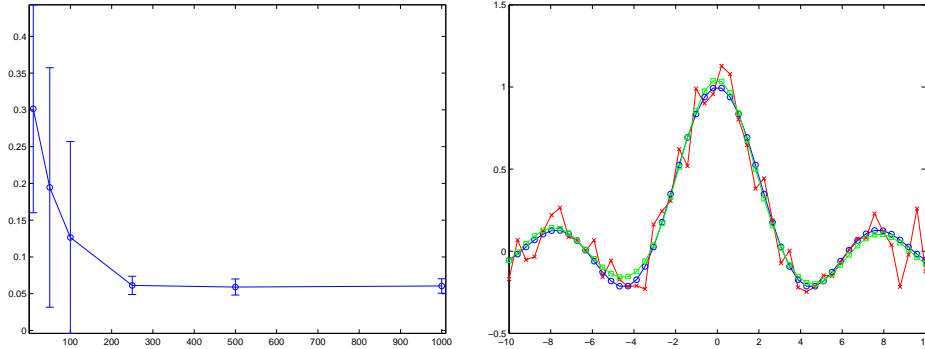

Figure 1: **Results for the sinc experiment**. Test error as a function of the number of samples (left), and example fit (right), showing the uncorrupted data (blue circles), noisy data (red crosses) and MMSE estimate (green squares). For this example the test error was 0.0309 and an average of 6.18 kernels were used.

| Method | Test Error | # Kernels | Noise Estimate |
|---|---|---|---|
| Figueiredo | 0.0455 | 7.0 | — |
| SVM | 0.0519 | 28.0 | — |
| RVM | 0.0494 | 6.9 | 0.0943 |
| VRVM | 0.0494 | 7.4 | 0.0950 |
| MCMC | 0.0468 | 6.5 | — |
| Sequential RVM | 0.0591 | 4.5 | 0.1136 |

Table 1: **Comparative performance results for the sinc data**. The batch results are reproduced from [1, 9].

### 4.2   Boston Housing Data

We also applied our algorithm to the popular Boston housing data set. We considered random train / test partitions of the data of size 300 / 206. We again used a Gaussian kernel, and set the width parameter to 5. For the model and algorithm parameters we used values similar to those for the sinc experiment, except for setting $\lambda = 5$ to allow a larger number of kernels. The results are summarised in Table 2. These were obtained by averaging over 10 random partitions of the data, and setting the number of samples to $N = 250$. The test error is comparable to those for the batch strategies, but far fewer kernels are required.

| Method | Test Error | # Kernels |
|---|---|---|
| SVM | 8.04 | 142.8 |
| RVM | 7.46 | 39.0 |
| Sequential RVM | 7.18 | 25.29 |

Table 2: **Comparative performance results for the Boston housing data**. The batch results are reproduced from [10].

## 5   Conclusions

In this paper we proposed a sequential estimation strategy for Bayesian kernel regression. Our algorithm is based on a generalisation of importance sampling, and incrementally updates a Monte Carlo representation of the target posterior distribution as more data points

become available. It achieves this through simple and intuitive model moves, reminiscent of the RJ-MCMC algorithm. It is further non-iterative, and requires only a single pass over the data, thus overcoming some of the computational difficulties associated with batch estimation strategies for kernel regression. Our algorithm is more general than the kernel regression problem considered here. Its application extends to any model for which the posterior can be evaluated up to a normalising constant. Initial experiments on two standard regression data sets showed our algorithm to compare favourably with existing batch estimation strategies for kernel regression.

### Acknowledgements

The authors would like to thank Mike Tipping for helpful comments during the experimental procedure. The work of Vermaak and Godsill was partially funded by QinetiQ under the project 'Extended and Joint Object Tracking and Identification', CU006-14890.

## References

[1] C. M. Bishop and M. E. Tipping. Variational relevance vector machines. In C. Boutilier and M. Goldszmidt, editors, *Proceedings of the 16th Conference on Uncertainty in Artificial Intelligence*, pages 46–53. Morgan Kaufmann, 2000.

[2] D. Crisan. Particle filters – a theoretical perspective. In A. Doucet, J. F. G. de Freitas, and N. J. Gordon, editors, *Sequential Monte Carlo Methods in Practice*, pages 17–38. Springer-Verlag, 2001.

[3] A. Doucet, J. F. G. de Freitas, and N. J. Gordon, editors. *Sequential Monte Carlo Methods in Practice*. Springer-Verlag, New York, 2001.

[4] N. J. Gordon, D. J. Salmond, and A. F. M. Smith. Novel approach to nonlinear/non-Gaussian Bayesian state estimation. *IEE Proceedings-F*, 140(2):107–113, 1993.

[5] P. J. Green. Reversible jump Markov chain Monte Carlo computation and Bayesian model determination. *Biometrika*, 82(4):711–732, 1995.

[6] G. Kitagawa. Monte Carlo filter and smoother for non-Gaussian nonlinear state space models. *Journal of Computational and Graphical Statistics*, 5(1):1–25, 1996.

[7] P. Del Moral and A. Doucet. Sequential Monte Carlo samplers. Technical Report CUED/F-INFENG/TR.443, Signal Processing Group, Cambridge University Engineering Department, 2002.

[8] R. M. Neal. Assessing relevance determination methods using DELVE. In C. M. Bishop, editor, *Neural Networks and Machine Learning*, pages 97–129. Springer-Verlag, 1998.

[9] S. S. Tham, A. Doucet, and R. Kotagiri. Sparse Bayesian learning for regression and classification using Markov chain Monte Carlo. In *Proceedings of the International Conference on Machine Learning*, pages 634–643, 2002.

[10] M. E. Tipping. The relevance vector machine. In S. A. Solla, T. K. Leen, and K. R. Müller, editors, *Advances in Neural Information Processing Systems*, volume 12, pages 652–658. MIT Press, 2000.

[11] M. E. Tipping. Sparse Bayesian learning and the relevance vector machine. *Journal of Machine Learning Research*, 1:211–244, 2001.

[12] M. E. Tipping and A. C. Faul. Fast marginal likelihood maximisation for sparse Bayesian models. In C. M. Bishop and B. J. Frey, editors, *Proceedings of the Ninth International Workshop on Artificial Intelligence and Statistics*, 2003.

[13] V. N. Vapnik. *Statistical Learning Theory*. John Wiley and Sons, New York, 1998.
